# A solvable connectionist model of immediate recall of ordered lists

**Neil Burgess**
Department of Anatomy, University College London
London WC1E 6BT, England
(e-mail: n.burgess@ucl.ac.uk)

## Abstract

A model of short-term memory for serially ordered lists of verbal stimuli is proposed as an implementation of the 'articulatory loop' thought to mediate this type of memory (Baddeley, 1986). The model predicts the presence of a repeatable time-varying 'context' signal coding the timing of items' presentation in addition to a store of phonological information and a process of serial rehearsal. Items are associated with context nodes and phonemes by Hebbian connections showing both short and long term plasticity. Items are activated by phonemic input during presentation and reactivated by context and phonemic feedback during output. Serial selection of items occurs via a winner-take-all interaction amongst items, with the winner subsequently receiving decaying inhibition. An approximate analysis of error probabilities due to Gaussian noise during output is presented. The model provides an explanatory account of the probability of error as a function of serial position, list length, word length, phonemic similarity, temporal grouping, item and list familiarity, and is proposed as the starting point for a model of rehearsal and vocabulary acquisition.

## 1 Introduction

Short-term memory for serially ordered lists of pronounceable stimuli is well described, at a crude level, by the idea of an 'articulatory loop' (AL). This postulates that information is phonologically encoded and decays within 2 seconds unless refreshed by serial rehearsal, see (Baddeley, 1986). It successfully accounts for (i)

the linear relationship between memory span $s$ (the number of items $s$ such that 50% of lists of $s$ items are correctly recalled) and articulation rate $r$ (the number of items that can be said per second) in which $s \approx 2r + c$, where $r$ varies as a function of the items, language and development; (ii) the fact that span is lower for lists of phonemically similar items than phonemically distinct ones; (iii) unattended speech and articulatory distractor tasks (e.g. saying blah-blah-blah...) both reduce memory span. Recent evidence suggests that the AL plays a role in the learning of new words both during development and during recovery after brain traumas, see e.g. (Gathercole & Baddeley, 1993). Positron emission tomography studies indicate that the phonological store is localised in the left supramarginal gyrus, whereas sub-vocal rehearsal involves Broca's area and some of the motor areas involved in speech planning and production (Paulesu et al., 1993).

However, the detail of the types of errors committed is not addressed by the AL idea. Principally: (iv) the majority of errors are 'order errors' rather than 'item errors', and tend to involve transpositions of neighbouring or phonemically similar items; (v) the probability of correctly recalling a list as a function of list length is a sigmoid; (vi) the probability of correctly recalling an item as a function of its serial position in the list (the 'serial position curve') has a bowed shape; (vii) span increases with the familiarity of the items used, specifically the $c$ in $s \approx 2r + c$ can increase from 0 to 2.5 (see (Hulme et al., 1991)), and also increases if a list has been previously presented (the 'Hebb effect'); (viii) 'position specific intrusions' occur, in which an item from a previous list is recalled at the same position in the current list. Taken together, these data impose strong functional constraints on any neural mechanism implementing the AL.

Most models showing serial behaviour rely on some form of 'chaining' mechanism which associates previous states to successive states, via recurrent connections of various types. Chaining of item or phoneme representations generates errors that are incompatible with human data, particularly (iv) above, see (Burgess & Hitch, 1992, Henson, 1994). Here items are maintained in serial order by association to a repeatable time-varying signal (which is suggested by position specific intrusions and is referred to below as 'context'), and by the recovery from suppression involved in the selection process – a modification of the 'competitive queuing' model for speech production (Houghton, 1990). The characteristics of STM for serially ordered items arise due to the way that context and phoneme information prompts the selection of each item.

## 2   The model

The model consists of 3 layers of artificial neurons representing context, phonemes and items respectively, connected by Hebbian connections with long and short term plasticity, see Fig. 1. There is a winner-take-all (WTA) interaction between item nodes: at each time step the item with the greatest input is given activation 1, and the others 0. The winner at the end of each time step receives a decaying inhibition that prevents it from being selected twice consecutively.

During **presentation**, phoneme nodes are activated by acoustic or (translated) visual input, activation in the context layer follows the pattern shown in Fig. 1, item nodes receive input from phoneme nodes via connections $w_{ij}$. Connections

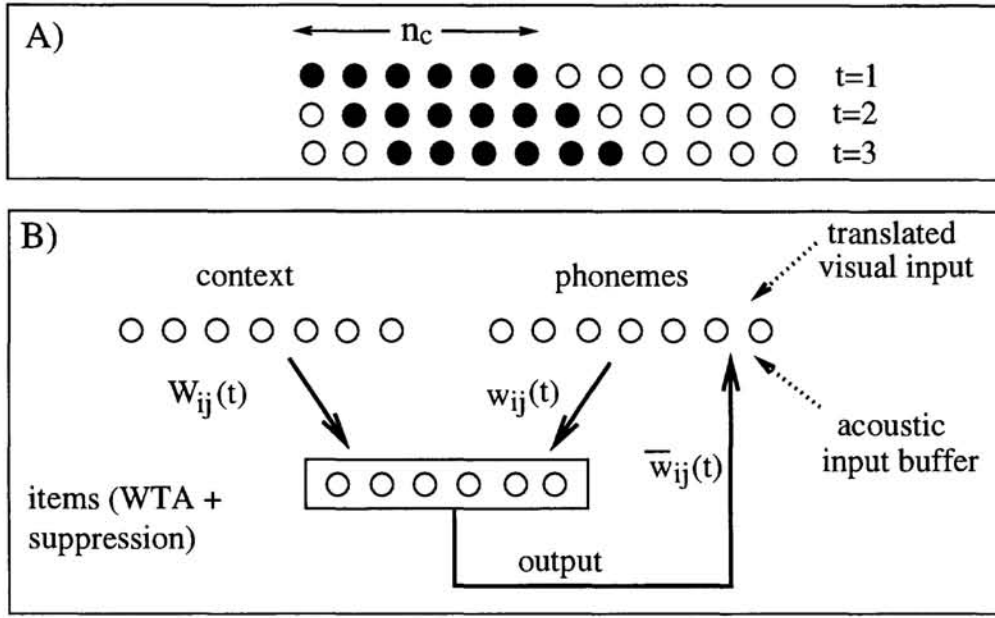

Figure 1: A) Context states as a function of serial position $t$; filled circles are active nodes, empty circles are inactive nodes. B) The architecture of the model. Full lines are connections with short and long term plasticity; dashed lines are routes by which information enters the model.

$W_{ij}(t)$ learn the association between the context state and the winning item, and $w_{ij}$ and $\bar{w}_{ij}$ learn the association with the active phonemes. During **recall**, the context layer is re-activated as in presentation, activation spreads to the item layer (via $W_{ij}(t)$) where one item wins and activates its phonemes (via $\bar{w}_{ij}(t)$). The item that now wins, given both context and phoneme inputs, is **output**, and then suppressed.

As described so far, the model makes no errors. Errors occur when Gaussian noise is added to items' activations during the selection of the winning item to be output. Errors are likely when there are many items with similar activation levels due to decay of connection weights and inhibition since presentation. Items may then be selected in the wrong order, and performance will decrease with the time taken to present or recall a list.

## 2.1   Learning and familiarity

Connection weights have both long and short term plasticity: $W_{ij}(t)$ (similarly $w_{ij}(t)$ and $\bar{w}_{ij}(t)$) have an incremental long term component $W_{ij}^{l}(t)$, and a one-shot short term component $W_{ij}^{s}(t)$ which decays by a factor $\Delta$ per second. The net weight of the connection is the sum of the two components: $W_{ij}(t) = W_{ij}^{l}(t) + W_{ij}^{s}(t)$. Learning occurs according to:

$$W_{ij}^{s}(t+1) = \begin{cases} c_j(t)a_i(t) & \text{if } c_j(t)a_i(t) > W_{ij}(t); \\ W_{ij}^{s}(t) & \text{otherwise,} \end{cases}$$

$$W_{ij}^{\ell}(t+1) \;=\; \begin{cases} W_{ij}^{\ell}(t) + \varepsilon c_j(t)a_i(t) & \text{if } c_j(t)a_i(t) > 0; \\ W_{ij}^{\ell}(t) & \text{otherwise,} \end{cases} \tag{1}$$

where $c_j(t)$ and $a_i(t)$ are the pre- and post-connection activations, and $\varepsilon$ decreases with $|W_{ij}^{\ell}(t)|$ so that the long term component saturates at some maximum value. These modifiable connection weights are never negative.

An item's 'familiarity' is reflected by the size of the long term components $w_{ij}^{\ell}$ and $\bar{w}_{ij}^{\ell}$ of the weights storing the association with its phonemes. These components increase with each (error-free) presentation or recall of the item. For lists of totally unfamiliar items, the item nodes are completely interchangeable having only the short-term connections $\bar{w}_{ij}^{s}$ to phoneme nodes that are learned at presentation. Whereas the presentation of a familiar item leads to the selection of a particular item node (due to the weights $w_{ij}^{\ell}$) and, during output, this item will activate its phonemes more strongly due to the weights $\bar{w}_{ij}^{\ell}$. Unfamiliar items that are phonemically similar to a familiar item will tend to be represented by the familiar item node, and can take advantage of its long-term item-phoneme weights $\bar{w}_{ij}^{\ell}$.

Presentation of a list leads to an increase in the long term component of the context-item association. Thus, if the same list is presented more than once its recall improves, and position specific intrusions from previous lists may also occur. Notice that only weights to or from an item winning at presentation or output are increased.

## 3   Details

There are $n_w$ items per list, $n_p$ phonemes per item, and a phoneme takes time $\ell_p$ seconds to present or recall. At time $t$, item node $i$ has activation $a_i(t)$, context node $i$ has activation $c_i(t)$, $C_t$ is the set of $n_c$ context nodes active at time $t$, phoneme node $i$ has activation $b_i(t)$ and $\mathcal{P}_i$ is the set of $n_p$ phonemes comprising item $i$.

Context nodes have activation 0 or $\sqrt{3/2n_c}$, phonemes take activation 0 or $1/\sqrt{n_p}$, so $W_{ij}^{s}(t) \le \sqrt{3/2n_c}$ and $w_{ij}^{s}(t) = \bar{w}_{ji}^{s}(t) \le 1/\sqrt{n_p}$, see (1). This sets the relative effect that context and phoneme layers have on items' activation, and ensures that items of neither few nor many phonemes are favoured, see (Burgess & Hitch, 1992). The long-term components of phoneme-item weights $w_{ij}^{\ell}(t)$ and $\bar{w}_{ji}^{\ell}(t)$ are $0.45/\sqrt{n_p}$ for familiar items, and $0.15/\sqrt{n_p}$ for unfamiliar items (chosen to match the data in Fig. 3B). The long-term components of context-item weights $W_{ij}^{\ell}(t)$ increase by $0.15/\sqrt{n_c}$ for each of the first few presentations or recalls of a list.

Apart from the WTA interaction, each item node $i$ has input:

$$h_i(t) = E_i(t) + I_i(t) + \eta_i, \tag{2}$$

where $I_i(t) < 0$ is a decaying inhibition imposed following an item's selection at presentation or output (see below), $\eta_i$ is a $(0, \sigma)$ Gaussian random variable added at output only, and $E_i(t)$ is the excitatory input to the item from the phoneme layer during presentation and the context and phoneme layers during recall:

$$E_i(t) = \begin{cases} \sum_j w_{ij}(t)b_j(t) & \text{during presentation;} \\ \sum_j W_{ij}(t)c_j(t) + w_{ij}(t)b_j(t) & \text{during recall.} \end{cases} \tag{3}$$

During recall phoneme nodes are activated according to $b_i(t) = \sum_j \bar{w}_{ij}(t)a_j(t)$.

One time step refers to the presentation or recall of an item and has duration $n_p \ell_p$. The variable $t$ increases by 1 per time step, and refers to both time and serial position. Short term connection weights and inhibition $I_i(t)$ decay by a factor $\Delta$ per second, or $\Delta^{n_p \ell_p}$ per time step.

The algorithm is as follows; rehearsal corresponds to repeating the recall phase.

Presentation
0. Set activations, inhibitions and short term weights to zero, $t = 1$.
1. Set the context layer to state $C_t : c_i(t) = \sqrt{3/2n_c}$ if $i \in C_t$; $c_i(t) = 0$ otherwise.
2. Input items, i.e. set the phoneme layer to state $\mathcal{P}_t : b_i(t) = 1/\sqrt{n_p}$ if $i \in \mathcal{P}_t$; $b_i(t) = 0$ otherwise.
3. Select the winning item, i.e. $a_k(t) = 1$ where $h_k(t) = \max_i \{h_i(t)\}$; $a_i(t) = 0$, for $i \neq k$.
4. Learning, i.e. increment all connection weights according to (1).
5. Decay, i.e. multiply short-term connection weights $W_{ij}^s(t)$, $w_{ij}^s(t)$ and $\bar{w}_{ij}^s(t)$, and inhibitions $I_i(t)$ by a factor $\Delta^{n_p \ell_p}$.
6. Inhibit winner, i.e. set $I_k(t) = -2$, where $k$ is the item selected in 3.
7. $t \to t + 1$, go to 1.

Recall
0. $t = 1$.
1. Set the context layer to state $C_t$, as above.
2. Set all phoneme activations to zero.
3. Select the winning item, as above.
4. Output. Activate phonemes via $\bar{w}_{ji}(t)$, select the winning item (in the presence of noise).
5. Learning, as above.
6. Decay, as above.
7. Inhibit winner, i.e. set $I_k(t) = -2$, where $k$ is the item selected in 4.
8. $t \to t + 1$, go to 1.

## 4  Analysis

The output of the model, averaged over many trials, depends on (i) the activation values of all items at the output step for each time $t$ and, (ii) given these activations and the noise level, the probability of each item being the winner. Estimation is necessary since there is no simple exact expression for (ii), and (i) depends on which items were output prior to time $t$.

I define $\gamma(t, i)$ to be the time elapsed, by output at time $t$, since item $i$ was last selected (at presentation or output), i.e. in the absence of errors:

$$\gamma(t, i) = \begin{cases} (t - i)\ell_p n_p & \text{if } i < t; \\ (n_w - (i - t))\ell_p n_p & \text{if } i \geq t. \end{cases} \tag{4}$$

If there have been no prior errors, then at time $t$ the inhibition of item $i$ is $I_i(t) = -2(\Delta)^{\gamma(t,i+1)}$, and short term weights to and from item $i$ have decayed by a factor $\Delta^{\gamma(t,i)}$. For a novel list of familiar items, the excitatory input to item $i$ during output at time $t$ is, see (3):

$$E_i(t) = 3\Delta^{\gamma(t,i)} ||C_i \cap C_t||/2n_c + (0.45 + \Delta^{\gamma(t,i)})^2 ||\mathcal{P}_i \cap \mathcal{P}_t||/n_p, \tag{5}$$

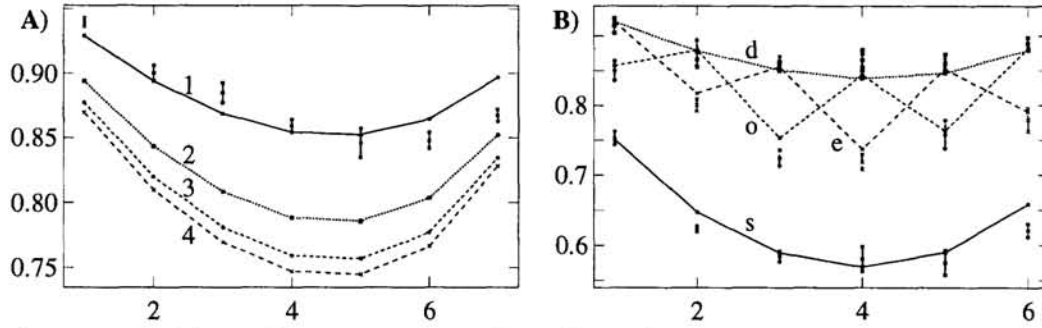

Figure 2: Serial position curves. Full lines show the estimation, extra markers are error bars at one standard deviation of 5 simulations of 1,000 trials each, see §5 for parameter values. A) Rehearsal. Four consecutive recalls of a list of 7 digits ('1',..,'4'). B) Phonemic similarity. SPCs are shown for lists of dissimilar letters ('d'), similar letters ('s'), and alternating similar and dissimilar letters with the similar ones in odd ('o') and even ('e') positions. C.f. (Baddeley, 1968, expt. V).

where $||\mathcal{X}||$ is the number of elements in set $\mathcal{X}$.

The probability $p(t, i)$ that item $i$ wins at time $t$ is estimated by the softmax function(Brindle, 1990):

$$p(t, i) \approx \frac{\exp\left(m_i(t)/\sigma'\right)}{\sum_{j=1}^{n_w} \exp\left(m_j(t)/\sigma'\right)},\tag{6}$$

where $m_i(t)$ is $h_i(t)$ without the noise term, see (2-3), and $\sigma' = 0.75\sigma$. For $\sigma = 0.5$ (the value used below), the r.m.s. difference between $p(t, i)$ estimated by simulation (500 trials) and by (6) is always less than 0.035 for $-1 < m_i(t) < 1$ with 2 to 6 items.

Which items have been selected prior to time $t$ affects $I_i(t)$ in $h_i(t)$ via $\gamma(t, i)$. $p(t, i)$ is estimated for all combinations of up to two prior errors using (6) with appropriate values of $m_i(t)$, and the average, weighted by the probability of each error combination, is used. The 'missing' probability corresponding to more than two prior errors is corrected for by normalising $p(t, i)$ so that $\sum_i p(t, i) = 1$ for $t = 1, .., n_w$. This overestimates the recency effect, especially in super-span lists.

## 5  Performance

The parameter values used are $\Delta = 0.75$, $n_c = 6$, $\sigma = 0.5$. Different types of item are modelled by varying $(n_p, \ell_p)$ : 'digits' correspond to $(2,0.15)$, 'letters' to $(2,0.2)$, and 'words' to $(5,0.15-0.3)$. 'Similar' items all have 1 phoneme in common, dissimilar items have none. Unless indicated otherwise, items are dissimilar and familiar, see §3 for how familiarity is modelled. The size of $\sigma$ relative to $\Delta$ is set so that digit span $\approx 7$. $n_p$ and $\ell_p$ are such that approximately 7 digits can be said in 2 seconds.

The model's performance is shown in Figs. 2 and 3. Fig. 2A: the increase in the long-term component of context-item connections during rehearsal brings stability after a small number of rehearsals, i.e. no further errors are committed. Fig. 2B: serial position curves show the correct effect of phonemic similarity among items.

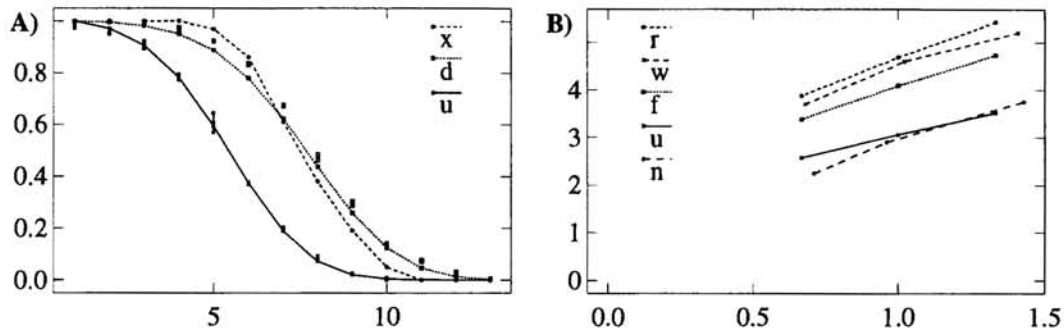

Figure 3: Item span. Full lines show the estimation, extra markers (A only) are error bars at one standard deviation of 3 simulations of 1,000 trials each, see §5 and §3 for parameter values. A) The probability of correctly recalling a whole list versus list length. Lists of digits ('d'), unfamiliar items (of the same length, 'u'), and experimental data on digits (adapted from (Guildford & Dallenbach, 1925), 'x') are shown. B) Span versus articulation rate (rate$= 1/\ell_p n_p$, with $n_p = 5$ and $\ell_p =0.15, 0.2$, and $0.3$). Calculated curves are shown for novel lists of familiar ('f') and unfamiliar ('u') words and lists of familiar words after 5 repetitions ('r'). Data on recall of words ('w') and non-words ('n') are also shown, adapted from (Hulme et al., 1991).

Fig. 3A: the probability of recalling a list correctly as a function of list length shows the correct sigmoidal relationship. Fig. 3B: item span shows the correct, approximately linear, relationship to articulation rate, with span for unfamiliar items below that for familiar items. Span increases with repeated presentations of a list in accordance with the 'Hebb effect'. Note that span is slightly overestimated for short lists of very long words.

## 5.1 Discussion and relation to previous work

This model is an extension of (Burgess & Hitch, 1992), primarily to model effects of rehearsal and item and list familiarity by allowing connection weights to show plasticity over different timescales, and secondly to show recency and phonemic similarity effects simultaneously by changing the way phoneme nodes are activated during recall. Note that the 'context' timing signal varies with serial position: reflecting the rhythm of presentation rather than absolute time (indeed the effect of temporal grouping can be modelled by modifying the context representations to reflect the presence of pauses during presentation (Hitch et al., 1995)), so presentation and recall rates cannot be varied.

The decaying inhibition that follows an items selection increases the locality of errors, i.e. if item $i + 1$ replaces item $i$, then item $i$ is most likely to replace item $i + 1$ in turn (rather than e.g. item $i + 2$). The model has two remaining problems: (i) selecting an item node to form the long term representation of a new item, without taking over existing item nodes, and (ii) learning the correct order of the phonemes within an item – a possible extension to address this problem is presented in (Hartley & Houghton, 1995).

The mechanism for selecting items is a modification of competitive queuing

(Houghton, 1990) in that the WTA interaction occurs at the item layer, rather than in an extra layer, so that only the winner is active and gets associated to context and phoneme nodes (this avoids partial associations of a context state to all items similar to the winner, which would prevent the zig-zag curves in Fig. 2B). The basic selection mechanism is sufficient to store serial order in itself, since items recover from suppression in the same order in which they were selected at presentation. The model maps onto the articulatory loop idea in that the selection mechanism corresponds to part of the speech production ('articulation') system and the phoneme layer corresponds to the 'phonological store', and predicts that a 'context' timing signal is also present. Both the phoneme and context inputs to the item layer serve to increase span, and in addition, the former causes phonemic similarity effects and the latter causes recency, position specific intrusions and temporal grouping effects.

## 6   Conclusion

I have proposed a simple mechanism for the storage and recall of serially ordered lists of items. The distribution of errors predicted by the model can be estimated mathematically and models a very wide variety of experimental data. By virtue of long and short term plasticity of connection weights, the model begins to address familiarity and the role of rehearsal in vocabulary acquisition. Many of the predicted error probabilities have not yet been checked experimentally: they are predictions. However, the major prediction of this model, and of (Burgess & Hitch, 1992), is that, in addition to a short-term store of phonological information and a process of sub-vocal rehearsal, STM for ordered lists of verbal items involves a third component which provides a repeatable time-varying signal reflecting the rhythm of the items' presentation.

**Acknowledgements:** I am grateful for discussions with Rik Henson and Graham Hitch regarding data, and with Tom Hartley and George Houghton regarding error probabilities, and to Mike Page for suggesting the use of the softmax function. This work was supported by a Royal Society University Research Fellowship.

## References

Baddeley A D (1968) *Quarterly Journal of Experimental Psychology* **20** 249-264.

Baddeley A D (1986) *Working Memory*, Clarendon Press.

Brindle, J S (1990) in: D S Touretzky (ed.) *Advances in Neural Information Processing Systems 2.* San Mateo, CA: Morgan Kaufmann.

Burgess N & Hitch G J (1992) *J. Memory and Language* **31** 429-460.

Gathercole S E & Baddeley A D (1993) *Working memory and language*, Erlbaum.

Guildford J P & Dallenbach K M (1925) *American J. of Psychology* **36** 621-628.

Hartley T & Houghton G (1995) *J. Memory and Language* to be published.

Henson R (1994) Tech. Report, M.R.C. Applied Psychology Unit, Cambridge, U.K.

Hitch G, Burgess N, Towse J & Culpin V (1995) *Quart. J. of Exp. Psychology*, submitted.

Houghton G (1990) in: R Dale, C Mellish & M Zock (eds.), *Current Research in Natural Language Generation* 287-319. London: Academic Press.

Hulme C, Maughan S & Brown G D A (1991) *J. Memory and Language* **30** 685-701.

Paulesu E, Frith C D & Frackowiak R S J (1993) *Nature* **362** 342-344.
